# Rule Induction through Integrated Symbolic and Subsymbolic Processing

Clayton McMillan, Michael C. Mozer, Paul Smolensky
Department of Computer Science and
Institute of Cognitive Science
University of Colorado
Boulder, CO 80309–0430

## Abstract

We describe a neural network, called *RuleNet*, that learns explicit, symbolic condition-action rules in a formal string manipulation domain. RuleNet discovers functional categories over elements of the domain, and, at various points during learning, extracts rules that operate on these categories. The rules are then injected back into RuleNet and training continues, in a process called *iterative projection*. By incorporating rules in this way, RuleNet exhibits enhanced learning and generalization performance over alternative neural net approaches. By integrating symbolic rule learning and subsymbolic category learning, RuleNet has capabilities that go beyond a purely symbolic system. We show how this architecture can be applied to the problem of case-role assignment in natural language processing, yielding a novel rule-based solution.

## 1 INTRODUCTION

We believe that neural networks are capable of more than pattern recognition; they can also perform higher cognitive tasks which are fundamentally rule-governed. Further we believe that they can perform higher cognitive tasks *better* if they incorporate rules rather than eliminate them. A number of well known cognitive models, particularly of language, have been criticized for going too far in eliminating rules in fundamentally rule-governed domains. We argue that with a suitable choice of high-level, rule-governed task, representation, processing architecture, and learning algorithm, neural networks can represent and learn rules involving higher-level categories while simultaneously learning those categories. The resulting networks can exhibit better learning and task performance than neural networks that do not incorporate rules, have capabilities that go beyond that of a purely symbolic rule-learning algorithm.

We describe an architecture, called *RuleNet*, which induces symbolic condition-action rules in a string mapping domain. In the following sections we describe this domain, the task and network architecture, simulations that demonstrate the potential for this approach, and finally, future directions of the research leading toward more general and complex domains.

## 2 DOMAIN

We are interested in domains that map input strings to output strings. A string consists of $n$ *slots*, each containing a symbol. For example, the string **abcd** contains the symbol **c** in slot 3. The domains we have studied are intrinsically rule-based, meaning that the mapping function from input to output strings can be completely characterized by explicit, mutually exclusive condition-action rules. These rules are of the general form "*if certain symbols are present in the input then perform a certain mapping from the input slots to the output slots.*" The conditions do not operate directly on the input symbols, but rather on *categories* defined over the input symbols. Input symbols can belong to multiple categories. For example, the words **boy** and **girl** are instances of the higher level category **HUMAN**. We denote instances with lowercase bold font, and categories with uppercase bold font. It should be apparent from context whether a letter string refers to a single instance, such as **boy**, or a string of instances, such as **abcd**.

Three types of conditions are allowed: 1) a *simple* condition, which states that an instance of some category must be present in a particular slot of the input string, 2) a *conjunction* of two simple conditions, and 3) a *disjunction* of two simple conditions. A typical condition might be that an instance of the category **W** must be present in slot 1 of the input string and an instance of category **Y** must be present in slot 3.

The action performed by a rule produces an output string in which the content of each slot is either a fixed symbol or a function of a particular input slot, with the additional constraint that each input slot maps to at most one output slot. In the present work, this function of the input slots is the identity function. A typical action might be to switch the symbols in slots 1 and 2 of the input, replace slot 3 with the symbol **a**, and copy slot 4 of the input to the output string unchanged, e.g., **abcd → baad**.

We call rules of this general form *second-order categorical permutation (SCP) rules*. The number of rules grows exponentially with the length of the strings and the number of input symbols. An example of an SCP rule for strings of length four is:

**if** (*input₁* is an instance of **W** and *input₃* is an instance of **Y**) **then**
($output_1$ = $input_2$, $output_2$ = $input_1$, $output_3$ = **a**, $output_4$=$input_4$)

where $input_\alpha$ and $output_\beta$ denote input slot $\alpha$ and output slot $\beta$, respectively. As a shorthand for this rule, we write [∧ **W_Y_ → 21a4**], where the square brackets indicate this is a rule, the "∧" denotes a conjunctive condition, and the "_" denotes a *wildcard* symbol. A disjunction is denoted by "∨".

This formal string manipulation task can be viewed as an abstraction of several interesting cognitive models in the connectionist literature, including case-role assignment (McClelland & Kawamoto, 1986), grapheme-phoneme mapping (Sejnowski & Rosenberg, 1987), and mapping verb stems to the past tense (Rumelhart & McClelland, 1986).

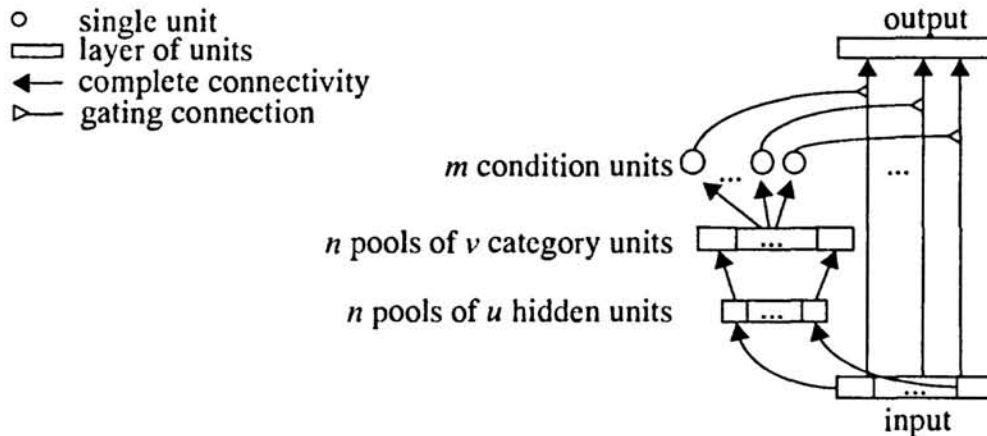

○    single unit
▭    layer of units
◀—  complete connectivity
▷—  gating connection

Figure 1: The RuleNet Architecture

## 3 TASK

RuleNet's task is to induce a compact set of rules that accurately characterizes a set of training examples. We generate training examples using a predefined rule base. The rules are over strings of length four and alphabets which are subsets of {a, b, c, d, e, f, g, h, i, j, k, l}. For example, the rule [v Y_W_→4h21] may be used to generate the exemplars:

$$\texttt{hedk} \rightarrow \texttt{kheh, cldk} \rightarrow \texttt{khlc, gbdj} \rightarrow \texttt{jhbg, gdbk} \rightarrow \texttt{khdg}$$

where category W consists of a, b, c, d, i, and category Y consists of f, g, h. Such exemplars form the corpus used to train RuleNet. Exemplars whose input strings meet the conditions of several rules are excluded. RuleNet's task is twofold: It must discover the categories solely based upon the usage of their instances, and it must induce rules based upon those categories.

The rule bases used to generate examples are minimal in the sense that no smaller set of rules could have produced the examples. Therefore, in our simulations the target number of rules to be induced is the same as the number used to generate the training corpus.

There are several traditional, symbolic systems, e.g., COBWEB (Fisher, 1987), that induce rules for classifying inputs based upon training examples. It seems likely that, given the correct representation, a system such as COBWEB could learn rules that would *classify* patterns in our domain. However, it is not clear whether such a system could also learn the action associated with each class. Classifier systems (Booker, et al., 1989) learn both conditions and actions, but there is no obvious way to map a symbol in slot $\alpha$ of the input to slot $\beta$ of the output. We have also devised a greedy combinatoric algorithm for inducing this type of rule, which has a number of shortcomings in comparison to RuleNet. See McMillan (1992) for comparisons of RuleNet and alternative symbolic approaches.

## 4 ARCHITECTURE

RuleNet can implement SCP rules of the type outlined above. As shown in Figure 1, RuleNet has five layers of units: an *input* layer, an *output* layer, a layer of *category* units, a layer of *condition* units, and a layer of *hidden* units. The operation of RuleNet can be divided into three functional components: categorization is performed in the mapping from the input layer to the category layer via the hidden units, the conditions are evaluated in the mapping from the category layer to the condition layer, and actions are performed in

the mapping from the input layer to the output layer, gated by the condition units.

The input layer is divided into $n$ pools of units, one for each slot, and activates the category layer, which is also divided into $n$ pools. Input pool $\alpha$ maps to category pool $\alpha$. Units in category pool $\alpha$ represent possible categorizations of the symbol in input slot $\alpha$. One or more category units will respond to each input symbol. The activation of the hidden and category units is computed with a logistic squashing function. There are $m$ units in the condition layer, one per rule. The activation of condition unit $i$, $p_i$, is computed as follows:

$$p_i = \frac{\text{logistic}(net_i)}{\sum_j \text{logistic}(net_j)}$$

The activation $p_i$ represents the probability that rule $i$ applies to the current input. The normalization enforces a soft winner-take-all competition among condition units. To the degree that a condition unit wins, it enables a set of weights from the input layer to the output layer. These weights correspond to the action for a particular rule. There is one set of weights, $A_i$, for each of the $m$ rules. The activation of the output layer, $\mathbf{y}$, is calculated from the input layer, $\mathbf{x}$, as follows:

$$\mathbf{y} = \sum_i^m p_i A_i \mathbf{x}$$

Essentially, the transformation $A_i$ for rule each rule $i$ is applied to the input, and it contributes to the output to the degree that condition $i$ is satisfied. Ideally, just one condition unit will be fully activated by a given input, and the rest will remain inactive.

This architecture is based on the local expert architecture of Jacobs, Jordan, Nowlan, and Hinton (1991), but is independently motivated in our work by the demands of the task domain. RuleNet has essentially the same structure as the Jacobs network, where the action substructure of RuleNet corresponds to their *local experts* and the condition substructure corresponds to their *gating network*. However, their goal—to minimize crosstalk between logically independent subtasks—is quite different than ours.

## 4.1 Weight Templates

In order to interpret the weights in RuleNet as symbolic SCP rules, it is necessary to establish a correspondence between regions of weight space and SCP rules.

A *weight template* is a parameterized set of constraints on some weights—a manifold in weight space—that has a direct correspondence to an SCP rule. The strategy behind iterative projection is twofold: constrain gradient descent so that weights stay close to templates in weight space, and periodically project the learned weights to the nearest template, which can then readily be interpreted as a set of SCP rules.

For SCP rules, there are three types of weight templates: one dealing with categorization, one with rule conditions, and one with rule actions. Each type of template is defined over a subset of the weights in RuleNet. The categorization templates are defined over the weights from input to category units, the condition templates are defined over the weights from category to condition units for each rule $i$, $c_i$, and the action templates are defined over the weights from input to output units for each rule $i$, $A_i$.

*Category templates.* The category templates specify that the mapping from each input slot $\alpha$ to category pool $\alpha$, for $1 \le \alpha \le n$, is uniform. This imposes category invariance across the input string.

*Condition templates.* The weight vector $c_i$, which maps category activities to the activity of condition unit $i$, has $vn$ elements—$v$ being the number of category units per slot and $n$ being the number of slots. The fact that the condition unit should respond to at most one category in each slot implies that at most one weight in each $v$-element subvector of $c_i$ should be nonzero. For example, assuming there are three categories, **W**, **X**, and **Y**, the vector $c_i$ that detects the simple condition *"input$_2$ is an instance of **X**"* is: (000 0$\varphi$0 000 000), where $\varphi$ is an arbitrary parameter. Additionally, a bias is required to ensure that the net input will be negative unless the condition is satisfied. Here, a bias value, $b$, of $-0.5\varphi$ will suffice. For disjunctive and conjunctive conditions, weights in *two* slots should be equal to $\varphi$, the rest zero, and the appropriate bias is $-.5\varphi$ or $-1.5\varphi$, respectively. There is a weight template for each condition type and each combination of slots that takes part in a condition. We generalize these templates further in a variety of ways. For instance, in the case where each input symbol falls into exactly one category, if a constant $\varepsilon_\alpha$ is added to all weights of $c_i$ corresponding to slot $\alpha$ and $\varepsilon_\alpha$ is also subtracted from $b$, the net input to condition unit $i$ will be unaffected. Thus, the weight template must include the $\{\varepsilon_\alpha\}$.

*Action templates.* If we wish the actions carried out by the network to correspond to the string manipulations allowed by our rule domain, it is necessary to impose some restrictions on the values assigned to the action weights for rule $i$, $A_i$. $A_i$ has an $n \times n$ block form, where $n$ is the length of input/output strings. Each block is a $k \times k$ submatrix, where $k$ is the number of elements in the representation of each input symbol. The block at block-row $\beta$, block-column $\alpha$ of $A_i$ copies *input$_\alpha$* to *output$_\beta$* if it is the identity matrix. Thus, the weight templates restrict each block to being either the identity matrix or the zero matrix. If *output$_\beta$* is to be a fixed symbol, then block-row $\beta$ must be all zero except for the output bias weights in block-row $\beta$.

The weight templates are defined over a submatrix $A_{i\beta}$, the set of weights mapping the input to an output slot $\beta$. There are $n+1$ templates, one for the mapping of each input slot to the output, and one for the writing of a fixed symbol to the output. An additional constraint that only one block may be nonzero in block-column $\alpha$ of $A_i$ ensures that *input$_\alpha$* maps to at most one output slot.

## 4.2 Constraints on Weight Changes

Recall that the strategy in iterative projection is to constrain weights to be close to the templates described above, in order that they may be readily interpreted as symbolic rules. We use a combination of hard and soft constraints, some of which we briefly describe here.

To ensure that during learning every block in $A_i$ approaches the identity or zero matrix, we constrain the off-diagonal terms to be zero and constrain weights along the diagonal of each block to be the same, thus limiting the degrees of freedom to one parameter within each block. All weights in $c_i$ except the bias are constrained to positive or zero values. Two soft constraints are imposed upon the network to encourage all-or-none categorization of input instances: A decay term is used on all weights in $c_i$ except the maximum in each slot, and a second cost term encourages binary activation of the category units.

## 4.3 Projection

The constraints described above do not guarantee that learning will produce weights that correspond exactly to SCP rules. However, using projection, it is possible to transform the condition and action weights such that the resulting network can be interpreted as rules. The essential idea of projection is to take a set of learned weights, such as $c_i$, and compute values for the parameters in each of the corresponding weight templates such that the resulting weights match the learned weights. The weight template parameters are estimated using a least squares procedure, and the closest template, based upon a Euclidean distance metric, is taken to be the projected weights.

## 5 SIMULATIONS

We ran simulations on 14 different training sets, averaging the performance of the network over at least five runs with different initial weights for each set. The training data were generated from SCP rule bases containing 2–8 rules and strings of length four. Between four and eight categories were used. Alphabets ranged from eight to 12 symbols. Symbols were represented by either local or distributed activity vectors. Training set sizes ranged from 3–15% of possible examples.

Iterative projection involved the following steps: (1) start with one rule (one set of $c_i$-$A_i$ weights), (2) perform gradient descent for 500-5,000 epochs, (3) project to the nearest set of SCP rules and add a new rule. Steps (2) and (3) were repeated until the training set was fully covered.

In virtually every run on each data set in which RuleNet converged to a set of rules that completely covered the training set, the rules extracted were exactly the original rules used to generate the training set. In the few remaining runs, RuleNet discovered an equivalent set of rules.

It is instructive to examine the evolution of a rule set. The rightmost column of Figure 2 shows a set of five rules over four categories, used to generate 200 exemplars, and the left portion of the Figure shows the evolution of the hypothesis set of rules learned by RuleNet over 20,000 training epochs, projecting every 4000 epochs. At epoch 8000, RuleNet has discovered two rules over two categories, covering 24.5% of the training set. At epoch 12,000, RuleNet has discovered three rules over three categories, covering 52% of the training set. At epoch 20,000, RuleNet has induced five rules over four categories that

| epoch 8000 | epoch 12,000 | epoch 20,000 | original rules/categ. |
|---|---|---|---|
| [∨ B_C_ → 4h21]<br>[∧ _B_C → 341f] | [∨ B_C_ → 4h21]<br>[∧ __EC → 2413]<br>[∧ _B_B → 321f] | [∨ B_C_ → 4h21]<br>[ _B__ → 4213]<br>[∨ _E_D → 342f]<br>[∧ _D_B → 3214]<br>[∨ __EC → 2413] | [∨ Y_W_ → 4h21]<br>[ _Y__ → 4213]<br>[∨ _Z_X → 342f]<br>[∧ _X_Y → 3214]<br>[∨ __ZW → 2413] |
| Categ.  Instance<br>B    f g h<br>C    a b c i | Categ.  Instance<br>B    f g h<br>C    a b c d i<br>E    a i j k | Categ.  Instance<br>C    a b c d i<br>D    e g l<br>B    f g h<br>E    a c i j k | Categ.  Instance<br>W    a b c d i<br>X    e g l<br>Y    f g h<br>Z    a c i j k |

Figure 2: Evolution of a Rule Set

Table 1: Generalization performance of RuleNet (average of five runs)

| | % of patterns correctly mapped | | | | | | | |
|---|---|---|---|---|---|---|---|---|
| Architecture | Data Set 1 (8 Rules) | | Data Set 2 (3 Rules) | | Data Set 3 (3 Rules) | | Data Set 4 (5 Rules) | |
| | train | test | train | test | train | test | train | test |
| RuleNet | 100 | 100 | 100 | 100 | 100 | 100 | 100 | 100 |
| Jacobs architecture | 100 | 22 | 100 | 7 | 100 | 14 | 100 | 27 |
| 3-layer backprop | 100 | 27 | 100 | 7 | 100 | 14 | 100 | 35 |
| # of patterns in set | 120 | 1635 | 45 | 1380 | 45 | 1380 | 75 | 1995 |

cover 100% of the training examples. A close comparison of these rules with the original rules shows that they only differ in the arbitrary labels RuleNet has attached to the categories.

Learning rules can greatly enhance generalization. In cases where RuleNet learns the original rules, it can be expected to generalize perfectly to any pattern created by those rules. We compared the performance of RuleNet to that of a standard three-layer backprop network (with 15 hidden units per rule) and a version of the Jacobs architecture, which in principle has the capacity to perform the task. Four rule bases were tested, and roughly 5% of the possible examples were used for training and the remainder were used for generalization testing. Outputs were thresholded to 0 or 1. The *cleaned up* outputs were compared to the targets to determine which were mapped correctly. All three learn the training set perfectly. However, on the test set, RuleNet's ability to generalize is 300% to 2000% better than the other systems (Table1).

Finally, we applied RuleNet to case-role assignment, as considered by McClelland and Kawamoto (1986). Case-role assignment is the problem of mapping syntactic constituents of a sentence to underlying semantic, or thematic, roles. For example, in the sentence, "The boy broke the window", *boy* is the subject at the syntactic level and the *agent*, or acting entity, at the semantic level. *Window* is the *object* at the syntactic level and the *patient*, or entity being acted upon, at the semantic level. The words of a sentence can be represented as a string of *n* slots, where each slot is labeled with a constituent, such as *subject*, and that slot is filled with the corresponding word, such as *boy*. The output is handled analogously. We used McClelland and Kawamoto's 152 sentences over 34 nouns and verbs as RuleNet's training set. The five categories and six rules induced by RuleNet are shown in Table 2, where S = subject, O = object, and wNP = noun in the *with* noun-phrase. We conjecture that RuleNet has induced such a small set of rules in part because it employs

Table 2: SCP Rules Induced by RuleNet in Case-Role Assignment

| Rule | Sample of Sentences Handled Correctly |
|---|---|
| if O = **VICTIM** then wNP→modifier | *The boy ate the pasta with cheese.* |
| if O = **THING** ∧ wNP = **UTENSIL** then wNP→instrument | *The boy ate the pasta with the fork.* |
| if S = **BREAKER** then S→instrument | *The rock broke the window.* |
| if S = **THING** then S→patient | *The window broke. The fork moved.* |
| if V = **moved** then **self**→patient | *The man moved.* |
| if S = **ANIMATE** then food→patient | *The lion ate.* |

implicit conflict resolution, automatically assigning strengths to categories and conditions. These rules cover 97% of the training set and perform the correct case-role assignments on 84% of the 1307 sentences in the test set.

# 6 DISCUSSION

RuleNet is but one example of a general methodology for rule induction in neural networks. This methodology involves five steps: 1) identify a fundamentally rule-governed domain, 2) identify a class of rules that characterizes that domain, 3) design a general architecture, 4) establish a correspondence between components of symbolic rules and manifolds of weight space—weight templates, and 5) devise a weight-template-based learning procedure.

Using this methodology, we have shown that RuleNet is able to perform both category and rule learning. Category learning strikes us as an intrinsically *subsymbolic* process. Functional categories are often fairly arbitrary (consider the classification of words as nouns or verbs) or have complex statistical structure (consider the classes "liberals" and "conservatives"). Consequently, real-world categories can seldom be described in terms of boolean (symbolic) expressions; subsymbolic representations are more appropriate.

While category learning is intrinsically subsymbolic, rule learning is intrinsically a symbolic process. The integration of the two is what makes RuleNet a unique and powerful system. Traditional symbolic machine learning approaches aren't well equipped to deal with subsymbolic learning, and connectionist approaches aren't well equipped to deal with the symbolic. RuleNet combines the strengths of each approach.

**Acknowledgments**

This research was supported by NSF Presidential Young Investigator award IRI-9058450, grant 90-21 from the James S. McDonnell Foundation, and DEC external research grant 1250 to MM; NSF grants IRI-8609599 and ECE-8617947 to PS; by a grant from the Sloan Foundation's computational neuroscience program to PS; and by the Optical Connectionist Machine Program of the NSF Engineering Research Center for Optoelectronic Computing Systems at the University of Colorado at Boulder.

**References**

Booker, L.B., Goldberg, D.E., and Holland, J.H. (1989). Classifier systems and genetic algorithms, *Artificial Intelligence* 40:235-282.

Fisher, D.H. (1987). Knowledge acquisition via incremental concept clustering. *Machine Learning* 2:139-172.

Jacobs, R., Jordan, M., Nowlan, S., Hinton, G. (1991). Adaptive mixtures of local experts. *Neural Computation*, 3:79-87.

McClelland, J. & Kawamoto, A. (1986). Mechanisms of sentence processing: assigning roles to constituents. In J.L. McClelland, D.E. Rumelhart, & the PDP Research Group, *Parallel Distributed Processing: Explorations in the microstructure of cognition, Vol. 2*. Cambridge, MA: MIT Press/Bradford Books.

McMillan, C. (1992). Rule induction in a neural network through integrated symbolic and subsymbolic processing. Unpublished Ph.D. Thesis. Boulder, CO: Department of Computer Science, University of Colorado.

Rumelhart, D., & McClelland, J. (1986). On learning the past tense of English verbs. In J.L. McClelland, D.E. Rumelhart, & the PDP Research Group, *Parallel Distributed Processing: Explorations in the microstructure of cognition.Vol. 2*. Cambridge, MA: MIT Press/Bradford Books.

Sejnowski, T. J. & Rosenberg, C. R. (1987). Parallel networks that learn to pronounce English text, *Complex Systems*, 1: 145-168.